# Bayesian Inference of Regular Grammar and Markov Source Models

**Kurt R. Smith and Michael I. Miller**
Biomedical Computer Laboratory
and
Electronic Signals and Systems Research Laboratory
Washington University, St. Louis, MO 63130

## ABSTRACT

In this paper we develop a Bayes criterion which includes the Rissanen complexity, for inferring regular grammar models. We develop two methods for regular grammar Bayesian inference. The first method is based on treating the regular grammar as a 1-dimensional Markov source, and the second is based on the combinatoric characteristics of the regular grammar itself. We apply the resulting Bayes criteria to a particular example in order to show the efficiency of each method.

## 1 MOTIVATION

We are interested in segmenting electron-microscope autoradiography (EMA) images by learning representational models for the textures found in the EMA image. In studying this problem, we have recognized that both structural and statistical features may be useful for characterizing textures. This has motivated us to study the source modeling problem for both structural sources and statistical sources. The statistical sources that we have examined are the class of one and two-dimensional Markov sources (see [Smith 1990] for a Bayesian treatment of Markov random field texture model inference), while the structural sources that we are primarily interested in here are the class of regular grammars, which are important due to the role that grammatical constraints may play in the development of structural features for texture representation.

# 2  MARKOV SOURCE INFERENCE

Our primary interest here is the development of a complete Bayesian framework for the process of inferring a regular grammar from a training sequence. However, we have shown previously that there exists a 1-D Markov source which generates the regular language defined via some regular grammar [Miller, 1988]. We can therefore develop a generalized Bayesian inference procedure over the class of 1-D Markov sources which enables us to learn the Markov source corresponding to the optimal regular grammar. We begin our analysis by developing the general structure for Bayesian source modeling.

## 2.1  BAYESIAN APPROACH TO SOURCE MODELING

We state the Bayesian approach to model learning: Given a set of source models $\{\theta_0, \theta_1, \cdots, \theta_{M-1}\}$ and the observation $x$, choose the source model $\theta_i$ which most accurately represents the unknown source that generated $x$. This decision is made by calculating Bayes risk over the possible models which produces a general decision criterion for the model learning problem:

$$\max_{\{\theta_0, \theta_1, \cdots, \theta_{M-1}\}} log\, P(x|\theta_i) + log\; P_i \,. \tag{2.1}$$

Under the additional assumption that the apriori probabilities over the candidate models are equivalent, the decision criterion becomes

$$\max_{\{\theta_0, \theta_1, \cdots, \theta_{M-1}\}} log\, P(x|\theta_i) \,, \tag{2.2}$$

which is the quantity that we will use in measuring the *accuracy* of a model's representation.

## 2.2  STOCHASTIC COMPLEXITY AND MODEL LEARNING

It is well known that when given finite data, Bayesian procedures of this kind which do not have any prior on the models suffer from the fundamental limitation that they will predict models of greater and greater complexity. This has led others to introduce priors into the Bayes hypothesis testing procedure based on the complexity of the model being tested [Rissanen, 1986]. In particular, for the Markov case the complexity is directly proportional to the number of transition probabilities of the particular model being tested with the prior exponentially decreasing with the associated complexity. We now describe the inclusion of the complexity measure in greater detail.

Following Rissanen, the basic idea is to uncover the model which assigns maximum probability to the observed data, while also being as simple as possible so as to require a small Kolmogorov description length. The complexity associated with a model having $k$ real parameters and a likelihood with $n$ independent samples, is the now well-known $\frac{k}{2}log\, n$ which allows us to express the generalization of the original Bayes procedure (2.2) as the quantity

$$\max_{\{\theta_0,\theta_1,\dots,\theta_{M-1}\}} \log P(x_n|\hat{\theta_i}) - \frac{k_{\theta_i}}{2} \log n \ . \tag{2.3}$$

Note well that $\hat{\theta_i}$ is the $k_{\theta_i}$-dimensional parameter parameterizing model $\theta_i$, which must be estimated from the observed data $x_n$. An alternative view of (2.3) is discovered by viewing the second term as the prior in the Bayes model (2.1) where the prior is defined as

$$P_{\theta_i} = e^{-\frac{k_{\theta_i}}{2} \log n} \ . \tag{2.4}$$

## 2.3 1-D MARKOV SOURCE MODELING

Consider that $x_n$ is a 1-D $n$-length string of symbols which is generated by an unknown finite-state Markov source. In examining (2.3), we recognize that for 1-D Markov sources $\log P(x|\hat{\theta_i})$ may be written as $\log \prod_{j=1}^{n-1} P_{\theta_i}(S(x_j)|S(x_{j-1}))$ where $S(x.)$ is a state function which evaluates to a state in the Markov source state set $S_{\theta_i}$. Using this notation, the Bayes hypothesis test for 1-D Markov sources may be expressed as:

$$\max_{\{\theta_0,\theta_1,\dots,\theta_{M-1}\}} \sum_{j=1}^{n-1} \log P_{\theta_i}(S(x_j)|S(x_{j-1})), \tag{2.5}$$

For the general Markov source inference problem, we know only that the string $x_n$ was generated by a 1-D Markov source, with the state set $S_{\theta_i}$ and the transition probabilities $P_{\theta_i}(S_k|S_l)$, $k,l \in S_{\theta_i}$ unknown. They must therefore be included in the inference procedure. To include the complexity term for this case, we note that the number of parameters to be estimated for model $\theta_i$ is simply the number of entries in the state-transition matrix $\hat{P}_{\theta_i}$, i.e. $k_{\theta_i} = |S_{\theta_i}|^2$. Therefore for 1-D Markov sources, the generalized Bayes hypothesis test including complexity may be stated as

$$\max_{\{\theta_0,\theta_1,\dots,\theta_{M-1}\}} \frac{1}{n} \sum_{j=1}^{n-1} \log \hat{P}_{\theta_i}(S(x_j)|S(x_{j-1})) - \frac{|S_{\theta_i}|^2}{2n} \log n, \tag{2.6}$$

where we have divided the entire quantity by $n$ in order to express the criterion in terms of bits per symbol. Note that a candidate Markov source model $\theta_i$ is initially specified by its order and corresponding state set $S_{\theta_i}$.

The procedure for inferring 1-D Markov source models can thus be stated as follows. Given a sequence $x_n$ from some unknown source, consider candidate Markov source models by computing the state function $S(x.)$ (determined by the candidate model order) over the entire string $x_n$. Enumerating the state transitions which occur in $x_n$ provides an estimate of the state-transition matrix $\hat{P}_{\theta_i}$ which is then used to compute (2.6). Now, the inferred Markov source becomes the one maximizing (2.6).

# 3 REGULAR GRAMMAR INFERENCE

Although the Bayes criterion developed for 1-D Markov sources (2.6) is a sufficient model learning criterion for the class of regular grammars, we will now show that by taking advantage of the apriori knowledge that the source is a regular grammar, the inference procedure can be made much more efficient. This apriori knowledge brings a special structure to the regular grammar inference problem in that not all allowable sets of Markov probabilities correspond to regular grammars. In fact, as shown in [Miller, 1988], corresponding to each regular grammar is a unique set of candidate probabilities, implying that the Bayesian solution which takes this into account will be far more efficient. We demonstrate that now.

## 3.1 BAYESIAN CRITERION USING GRAMMAR COMBINATORICS

Our approach is to use the combinatoric properties of the regular grammar in order to develop the optimal Bayes hypothesis test. We begin by defining the regular grammar.

**Definition:** A regular grammar $G$ is a quadruple $\langle V_N, V_T, S_S, R \rangle$ where $V_N, V_T$ are finite sets of non-terminal symbols (or states) and terminal symbols respectively, $S_S$ is the sentence start state, and $R$ is a finite set of production rules consisting of the transformation of a non-terminal symbol to either a terminal followed by a non-terminal, or a terminal alone, i.e.,

$$S_i \rightarrow W_j S_k \ or \ S_i \rightarrow W_j, \quad where \ W_j \in V_T, \ S_i, S_k \in V_N .$$

In the class of regular grammars that we consider, we define the *depth* of the language as the maximum number of terminal symbols which make up a nonterminal symbol. Corresponding to each regular grammar is an associated incidence matrix $B$ with the $i,k^{th}$ entry $B_{i,k}$ equal to the number of times there is a production for some terminal $j$ and non-terminals $i,k$ of the form $S_i \rightarrow W_j S_k \in R$. Also associated with each grammar $G_i$ is the set of all $n$-length strings produced by the grammar, denoted as the regular language $\mathcal{L}_n(G_i)$.

Now we make the quite reasonable assumption that no string in the language $\mathcal{L}_n(G_i)$ is more or less probable apriori than any other string in that language. This indicates that all $n$-length strings that can be generated by $G_i$ are equiprobable with a probability dictated by the combinatorics of the language as

$$P(x_n | G_i) = \frac{1}{|\mathcal{L}_n(G_i)|} , \tag{3.1}$$

where $|\mathcal{L}_n(G_i)|$ denotes the number of $n$-length sequences in the language which can be computed by considering the combinatorics of the language as follows:

$$|\mathcal{L}_n(G_i)| = \lambda_{G_i}^n ,$$

with $\lambda_{G_i}$ corresponding to the largest eigenvalue of the state-transition matrix $B_{G_i}$. This results from the combinatoric growth rate being determined by the sum of the entries in the $n^{th}$ power state-transition matrix $B_{G_i}^n$, which grows as the largest eigenvalue $\lambda_{G_i}$ of $B_{G_i}$ [Blahut, 1987]. We can now write (3.1) in these terms as

$$P(x_n|G_i) = \lambda_{G_i}^{-n}, \tag{3.2}$$

which expresses the probability of the sequence $x_n$ in terms of the combinatorics of $G_i$.

We now use this combinatoric interpretation of the probability to develop Bayes decision criterion over two candidate grammars. Assume that there exists a finite space of sequences $X$, all of which may be generated by one of the two possible grammars $\{G_0, G_1\}$. Now by dividing this observation space $X$ into two decision regions, $X_0$ (for $G_0$) and $X_1$ (for $G_1$), we can write Bayes risk $R$ in terms of the observation probabilities $P(x_n|G_0), P(x_n|G_1)$:

$$R = \sum_{x_n \in X_1} P(x_n|G_0) + \sum_{x_n \in X_0} P(x_n|G_1). \tag{3.3}$$

This implementation of Bayes risk assumes that sequences from each grammar occur equiprobably apriori and that the cost of choosing the incorrect grammar is equal to 1. Now incorporating the combinatoric counting probabilities (3.2), we can rewrite (3.3) as

$$R = \sum_{x_n \in X_1} \lambda_{G_0}^{-n} + \sum_{x_n \in X_0} \lambda_{G_1}^{-n}$$

which can be rewritten

$$R = \frac{1}{2} + \sum_{x_n \in X_0} \left\{ \lambda_{G_1}^{-n} - \lambda_{G_0}^{-n} \right\}. \tag{3.4}$$

The risk is therefore minimized by choosing $G_0$ if $\lambda_{G_1}^{-n} < \lambda_{G_0}^{-n}$ and $G_1$ if $\lambda_{G_1}^{-n} > \lambda_{G_0}^{-n}$. This establishes the *likelihood ratio* for the grammar inference problem:

$$\frac{\lambda_{G_1}^{-n}}{\lambda_{G_0}^{-n}} \underset{\substack{< \\ G_0}}{\overset{\substack{G_1 \\ >}}{}} 1,$$

which can alternatively be expressed in terms of the *log* as

$$\underset{\{G_0, G_1\}}{max} - n \log \lambda_{G_i}.$$

Recognizing this as the *maximum likelihood* decision, this decision criterion is easily generalized to M hypothesis. Now by ignoring any complexity component, the generalized Bayes test for a regular grammar can be stated as

$$\underset{\{G_0,G_1,...,G_{M-1}\}}{max} - n \log \lambda_{G_i}, \tag{3.5}$$

where $\lambda_{G_i}$ is the largest eigenvalue of the estimated incidence matrix $\widehat{B}_{G_i}$ corresponding to grammar $G_i$ where $\widehat{B}_{G_i}$ is estimated from $x_n$.

The complexity factor to be included in this Bayesian criterion differs from the complexity term in (2.3) due to the fact that the parameters to be estimated are now the entries in the $\widehat{B}_{G_i}$ matrix which are strictly binary. From a description length interpretation then, these parameters can be fully described using 1 bit per entry in $\widehat{B}_{G_i}$. The complexity term is thus simply $|S_{G_i}|^2$ which now allows us to write the Bayes inference criterion for regular grammars as

$$\underset{\{G_0,G_1,...,G_{M-1}\}}{max} - \log \lambda_{G_i} - \frac{|S_{G_i}|^2}{n}, \tag{3.6}$$

in terms of bits per symbol. We can now state the algorithm for inferring grammars.

### Regular Grammar Inference Algorithm

1. Initialize the grammar depth to $d=1$.

2. Compute $|S_G| = |V_T|^d$.

3. Using the state function $S_d(x_i)$ corresponding to the current depth, compute the state transitions at all sites $x_i$ in the observed sequence $x_n$ in order to estimate the incidence matrix $\widehat{B}_{G_i}$ for the grammar currently being considered.

4. Compute $\lambda_{G_i}$ from $\widehat{B}_{G_i}$. (recall that this is the largest eigenvalue of $\widehat{B}_{G_i}$).

5. Using $\lambda_{G_i}$ and $|S_{G_i}|$ compute (3.6) - denote this as $I_{G_i} = - \log \lambda_{G_i} - \frac{|S_{G_i}|^2}{n}$.

6. Increase the grammar depth $d=d+1$ and goto 2 (i.e. test another candidate grammar) until $I_{G_i}$ discontinues to increase.

The regular grammar of minimum depth which maximizes $I_{G_i}$ (i.e. maximizes (3.6)) is then the optimal regular grammar source model for the given sequence $x_n$.

### 3.2 REGULAR GRAMMAR INFERENCE RESULTS

To compare the efficiency of the two Bayes criteria (2.6) and (3.6), we will consider a regular grammar inference experiment. The regular grammar that we will attempt to learn, which we refer to as the 4-0,1s regular grammar, is a run-length constrained binary

grammar which disallows 4 consecutive occurrences of a 0 or a 1. Referring to the regular grammar definition, we note that this regular grammar can be described by its incidence matrix

$$B_{4\text{-}0,1} = \begin{bmatrix} 0 & 0 & 0 & 1 & 0 & 0 \\ 1 & 0 & 0 & 1 & 0 & 0 \\ 0 & 1 & 0 & 1 & 0 & 0 \\ 0 & 0 & 1 & 0 & 1 & 0 \\ 0 & 0 & 1 & 0 & 0 & 1 \\ 0 & 0 & 1 & 0 & 0 & 0 \end{bmatrix},$$

where the states corresponding to row and column indices are

$$S_1 = 000, \ S_2 = 00, S_3 = 0, S_4 = 1, S_5 = 11, S_6 = 111 \ .$$

Note that this regular grammar has a depth equal to 3 and thus the corresponding Markov source has an order equal to 3.

The inference experiment may be described as follows. Given a training set of length 16 strings from the 4-0,1s language, we apply the Bayes criteria (2.6) and (3.6) in an attempt to infer the regular grammar in each case. We compute the criteria for five candidate models of order/depth 1 through 5 (recall that this defines the size of the state set for the Markov source and the regular grammar, respectively).

Treating the unknown regular grammar as a Markov source, we estimate the corresponding state-transition matrix $\widehat{P}$ and then compute the Bayes criterion according to (2.6) for each of the five candidate models. We compute the criterion as a function of the number of training samples for each candidate model and plot the result in Figure 1a.

Similarly, we estimate the incidence matrix $\widehat{B}$ and compute the Bayes criterion according to (3.6) for each of the five regular grammar candidate models, and plot the results as a function of the number of training samples in Figure 1b.

We compare the two Bayesian criteria by examining Figures 1a and 1b. Note that criterion (3.6) discovers the correct regular grammar (depth = 3) after only 50 training samples (Figure 1b), while the equivalent Markov source (order = 3) is found only after almost 500 training samples have been used in computing (2.6) (Figure 1a). This points out that a much more efficient inference procedure exists for regular grammars by taking advantage of the apriori grammar information (i.e. only the depth and the binary incidence matrix $\widehat{B}$ must be estimated), whereas for 1-D Markov sources, both the order and the real-valued state-transition matrix $\widehat{P}$ must be estimated.

## 4. CONCLUSION

In conclusion, we stress the importance of casting the source modeling problem within a Bayesian framework which incorporates priors based on the model complexity and known model attributes. Using this approach, we have developed an efficient Bayesian

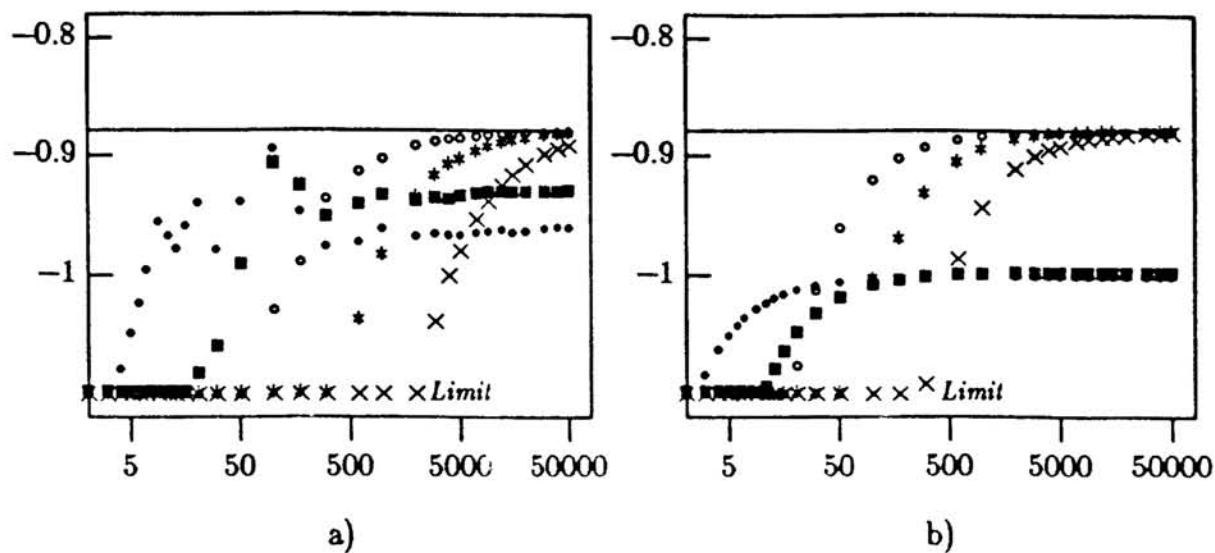

a)                                    b)

Grammar depth $d$ /Markov order: $\bullet = 1, \blacksquare = 2, \circ = 3, * = 4, \times = 5$ .

**Figure 1:** Results of computing Bayes criterion measures (2.6) and (3.6) vs. the number of training samples - a) Markov source criterion (2.6); b) Regular grammar combinatoric criterion (3.6).

framework for inferring regular grammars. This type of Bayesian model is potentially quite useful for the texture analysis and image segmentation problem where a consistent framework is desired for considering both structural and statistical features in the texture/image representation.

**Acknowledgements**

This research was supported by the NSF via a Presidential Young Investigator Award ECE-8552518 and by the NIH via a DRR Grant RR-1380.

**References**

Blahut, R. E. (1987), *Principles and Practice of Information Theory* , Addison-Wesley Publishing Co., Reading, MA.

Miller, M. I., Roysam, B, Smith, K. R., and Udding, J. T (1988), "Mapping Rule-Based Regular Grammars to Gibbs Distributions", *AMS-IMS-SIAM Joint Conference on SPATIAL STATISTICS AND IMAGING*, American Mathematical Society.

Rissanen, J. (1986), "Stochastic Complexity and Modeling", *Annals of Statistics*, 14, no.3, pp. 1080-1100.

Smith, K. R., Miller, M. I. (1990), "A Bayesian Approach Incorporating Rissanen Complexity for Learning Markov Random Field Texture Models", Proceedings of Int. Conference on Acoustics, Speech, and Signal Processing, Albuquerque, NM.